# Adaptive Nonlinear System Identification with Echo State Networks

**Herbert Jaeger**
International University Bremen
D-28759 Bremen, Germany
*h.jaeger@iu-bremen.de*

## Abstract

Echo state networks (ESN) are a novel approach to recurrent neural network training. An ESN consists of a large, fixed, recurrent "reservoir" network, from which the desired output is obtained by training suitable output connection weights. Determination of optimal output weights becomes a linear, uniquely solvable task of MSE minimization. This article reviews the basic ideas and describes an online adaptation scheme based on the RLS algorithm known from adaptive linear systems. As an example, a 10-th order NARMA system is adaptively identified. The known benefits of the RLS algorithms carry over from linear systems to nonlinear ones; specifically, the convergence rate and misadjustment can be determined at design time.

## 1 Introduction

It is fair to say that difficulties with existing algorithms have so far precluded supervised training techniques for recurrent neural networks (RNNs) from widespread use. *Echo state networks* (ESNs) provide a novel and easier to manage approach to supervised training of RNNs. A large (order of 100s of units) RNN is used as a "reservoir" of dynamics which can be excited by suitably presented input and/or fed-back output. The connection weights of this reservoir network are not changed by training. In order to compute a desired output dynamics, only the weights of connections from the reservoir to the output units are calculated. This boils down to a linear regression. The theory of ESNs, references and many examples can be found in [5] [6]. A tutorial is [7]. A similar idea has recently been independently investigated in a more biologically oriented setting under the name of "liquid state networks" [8] [9].

In this article I describe how ESNs can be conjoined with the "recursive least squares" (RLS) algorithm, a method for fast online adaptation known from linear systems. The resulting RLS-ESN is capable of tracking a 10-th order nonlinear system with high quality in convergence speed and residual error. Furthermore, the approach yields apriori estimates of tracking performance parameters and thus allows one to design nonlinear trackers according to specifications[1].

*Article organization.* Section 2 recalls the basic ideas and definitions of ESNs and introduces an augmentation of the basic technique. Section 3 demonstrates ESN offline learning on the 10th order system identification task. Section 4 describes the principles of using the RLS algorithm with ESN networks and presents a simulation study. Section 5 wraps up.

## 2 Basic ideas of echo state networks

For the sake of a simple notation, in this article I address only single-input, single-output systems (general treatment in [5]). We consider a discrete-time "reservoir" RNN with $N$ internal network units, a single extra input unit, and a single extra output unit. The input at time $n \geq 1$ is $u(n)$, activations of internal units are $\mathbf{x}(n) = (x_1(n), \ldots, x_N(n))$, and activation of the output unit is $y(n)$. Internal connection weights are collected in an $N \times N$ matrix $\mathbf{W} = (w_{ij})$, weights of connections going from the input unit into the network in an $N$-element (column) weight vector $\mathbf{w}^{in} = (w_i^{in})$, and the $N+1$ (input-and-network)-to-output connection weights in a $N+1$-element (row) vector $\mathbf{w}^{out} = (w_i^{out})$. The output weights $\mathbf{w}^{out}$ will be learned, the internal weights $\mathbf{W}$ and input weights $\mathbf{w}^{in}$ are fixed before learning, typically in a sparse random connectivity pattern. Figure 1 sketches the setup used in this article.

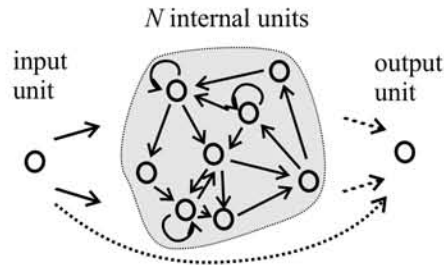

$N$ internal units

input unit

output unit

Figure 1: Basic setup of ESN. Solid arrows: fixed weights; dashed arrows: trainable weights.

The activation of internal units and the output unit is updated according to

$$
\begin{aligned}
\mathbf{x}(n+1) &= \mathbf{f}(\mathbf{W}\mathbf{x}(n) + \mathbf{w}^{in}u(n+1) + \nu(n+1)) & (1)\\
y(n+1) &= f^{out}(\mathbf{w}^{out}(u(n+1), \mathbf{x}(n+1))), & (2)
\end{aligned}
$$

where $\mathbf{f}$ stands for an element-wise application of the unit nonlinearity, for which we here use tanh; $\nu(n+1)$ is an optional noise vector; $(u(n+1), \mathbf{x}(n+1))$ is a vector concatenated from $u(n+1)$ and $\mathbf{x}(n+1)$; and $f^{out}$ is the output unit's non-linearity (*tanh* will be used here, too). Training data is a stationary I/O signal $(u_{teach}(n), y_{teach}(n))$. When the network is updated according to (1), then under certain conditions the network state becomes asymptotically independent of initial conditions. More precisely, if the network is started from two arbitrary states $\mathbf{x}(0), \tilde{\mathbf{x}}(0)$ and is run with the same input sequence in both cases, the resulting state sequences $\mathbf{x}(n), \tilde{\mathbf{x}}(n)$ converge to each other. If this condition holds, the reservoir network state will asymptotically depend only on the input history, and the network

tained in a tutorial Mathematica notebook which can be fetched from http://www.ais.fraunhofer.de/INDY/ESNresources.html.

is said to be an *echo state network* (ESN). A sufficient condition for the echo state property is contractivity of $\mathbf{W}$. In practice it was found that a weaker condition suffices, namely, to ensure that the spectral radius $|\lambda_{\max}|$ of $\mathbf{W}$ is less than unity. [5] gives a detailed account.

Consider the task of computing the output weights such that the teacher output is approximated by the network. In the ESN approach, this task is spelled out concretely as follows: compute $\mathbf{w}^{\text{out}}$ such that the training error

$$\epsilon_{\text{train}}(n) = (f^{\text{out}})^{-1} y_{\text{teach}}(n) - \mathbf{w}^{\text{out}}(u_{\text{teach}}(n), \mathbf{x}(n)) \qquad (3)$$

is minimized in the mean square sense. Note that the effect of the output non-linearity is undone by $(f^{\text{out}})^{-1}$ in this error definition. We dub $(f^{\text{out}})^{-1} y_{\text{teach}}(n)$ the teacher *pre-signal* and $(f^{\text{out}})^{-1}(\mathbf{w}^{\text{out}}(u_{\text{teach}}(n), \mathbf{x}(n)) + \nu(n))$ the network's *pre-output*. The computation of $\mathbf{w}^{\text{out}}$ is a linear regression. Here is a sketch of an offline algorithm for the entire learning procedure:

1. Fix a RNN with a single input and a single output unit, scaling the weight matrix $\mathbf{W}$ such that $|\lambda_{\max}| < 1$ obtains.
2. Run this RNN by driving it with the teaching input signal. Dismiss data from initial transient and collect remaining input+network states $(u_{\text{teach}}(n), \mathbf{x}_{\text{teach}}(n))$ row-wise into a matrix $\mathbf{M}$. Simultaneously, collect the remaining training pre-signals $(f^{\text{out}})^{-1} y_{\text{teach}}(n)$ into a column vector $\mathbf{r}$.
3. Compute the pseudo-inverse $\mathbf{M}^{-1}$, and put $\mathbf{w}^{\text{out}} = (\mathbf{M}^{-1}\mathbf{r})^{\top}$ (where $\cdot^{\top}$ denotes transpose).
4. Write $\mathbf{w}^{\text{out}}$ into the output connections; the ESN is now trained.

The modeling power of an ESN grows with network size. A cheaper way to increase the power is to use additional nonlinear transformations of the network state $\mathbf{x}(n)$ for computing the network output in (2). We use here a squared version of the network state. Let $\mathbf{w}^{\text{out}}_{\text{squares}}$ denote a length $2N + 2$ output weight vector and $\mathbf{x}_{\text{squares}}(n)$ the length $2N+2$ (column) vector $(u(n), x_1(n), \ldots, x_N(n), u^2(n), x_1^2(n), \ldots, x_N^2(n))$. Keep the network update (1) unchanged, but compute outputs with the following variant of (2):

$$y(n + 1) \quad = \quad f^{\text{out}}(\mathbf{w}^{\text{out}}_{\text{squares}} \mathbf{x}_{\text{squares}}(n + 1)). \qquad (4)$$

The "reservoir" and the input is now tapped by linear and quadratic connections. The learning procedure remains linear and now goes like this:

1. (unchanged)
2. Drive the ESN with the training input. Dismiss initial transient and collect remaining augmented states $\mathbf{x}_{\text{squares}}(n)$ row-wise into $\mathbf{M}$. Simultaneously, collect the training pre-signals $(f^{\text{out}})^{-1} y_{\text{teach}}(n)$ into a column vector $\mathbf{r}$.
3. Compute the pseudo-inverse $\mathbf{M}^{-1}$, and put $\mathbf{w}^{\text{out}}_{\text{squares}} = (\mathbf{M}^{-1}\mathbf{r})^{\top}$.
4. The ESN is now ready for exploitation, using output formula (4).

## 3 Identifying a 10th order system: offline case

In this section the workings of the augmented algorithm will be demonstrated with a nonlinear system identification task. The system was introduced in a survey-and-unification-paper [1]. It is a 10th-order NARMA system:

$$d(n + 1) = 0.3\,d(n) + 0.05\,d(n) \left[ \sum_{i=0}^{9} d(n - i) \right] + 1.5\,u(n - 9)\,u(n) + 0.1. \qquad (5)$$

*Network setup.* An $N = 100$ ESN was prepared by fixing a random, sparse connection weight matrix $\mathbf{W}$ (connectivity 5 %, non-zero weights sampled from uniform distribution in $[-1, 1]$, the resulting raw matrix was re-scaled to a spectral radius of 0.8, thus ensuring the echo state property). An input unit was attached with a random weight vector $\mathbf{w}^{\text{in}}$ sampled from a uniform distribution over $[-0.1, 0.1]$.

*Training data and training.* An I/O training sequence was prepared by driving the system (5) with an i.i.d. input sequence sampled from the uniform distribution over $[0, 0.5]$, as in [1]. The network was run according to (1) with the training input for 1200 time steps with uniform noise $\nu(n)$ of size 0.0001. Data from the first 200 steps were discarded. The remaining 1000 network states were entered into the augmented training algorithm, and a 202-length augmented output weight vector $\mathbf{w}^{\text{out}}_{\text{squares}}$ was calculated.

*Testing.* The learnt output vector was installed and the network was run from a zero starting state with newly created testing input for 2200 steps, of which the first 200 were discarded. From the remaining 2000 steps, the NMSE test error $\text{NMSE}_{\text{test}} = \frac{E[(y(n) - d(n))^2]}{\sigma^2(d)}$ was estimated. A value of $\text{NMSE}_{\text{test}} \approx 0.032$ was found.

*Comments.* (1) The noise term $\nu(n)$ functions as a regularizer, slightly compromising the training error but improving the test error. (2) Generally, the larger an ESN, the more training data is required and the more precise the learning. Set up exactly like in the described 100-unit example, an augmented 20-unit ESN trained on 500 data points gave $\text{NMSE}_{\text{test}} \approx 0.31$, a 50-unit ESN trained on 1000 points gave $\text{NMSE}_{\text{test}} \approx 0.084$, and a 400-unit ESN trained on 4000 points gave $\text{NMSE}_{\text{test}} \approx 0.0098$.

*Comparison.* The best NMSE *training* [!] error obtained in [1] on a length 200 training sequence was $\text{NMSE}_{\text{train}} \approx 0.241$[2] However, the level of precision reported [1] and many other published papers about RNN training appear to be based on suboptimal training schemes. After submission of this paper I went into a friendly modeling competition with Danil Prokhorov who expertly applied EKF-BPPT techniques [3] to the same tasks. His results improve on [1] results by an order of magnitude and reach a slightly better precision than the results reported here.

## 4 Online adaptation of ESN network

Because the determination of optimal (augmented) output weights is a linear task, standard recursive algorithms for MSE minimization known from adaptive linear signal processing can be applied to online ESN estimation. I assume that the reader is familiar with the basic idea of FIR tap-weight (Wiener) filters: i.e., that $N$ input signals $x_1(n), \ldots, x_N(n)$ are transformed into an output signal $y(n)$ by an inner product with a tap-weight vector $(w_1, \ldots, w_N)$: $y(n) = w_1 x_1(n) + \ldots + w_N x_N(n)$. In the ESN context, the input signals are the $2N + 2$ components of the augmented input+network state vector, the tap-weight vector is the augmented output weight vector, and the output signal is the network pre-output $(f^{\text{out}})^{-1} y(n)$.

## 4.1 A refresher on adaptive linear system identification

For a recursive online estimation of tap-weight vectors, "recursive least squares" (RLS) algorithms are widely used in linear signal processing when fast convergence is of prime importance. A good introduction to RLS is given in [2], whose notation I follow. An online algorithm in the augmented ESN setting should do the following: given an open-ended, typically non-stationary training I/O sequence $(u_{\text{teach}}(n), y_{\text{teach}}(n))$, at each time $n \geq 1$ determine an augmented output weight vector $\mathbf{w}_{\text{squares}}^{\text{out}}(n)$ which yields a good model of the current teacher system.

Formally, an RLS algorithm for ESN output weight update minimizes the exponentially discounted square "pre-error"

$$\sum_{k=1}^{n} \lambda^{n-k} \left( (f^{\text{out}})^{-1} y_{\text{teach}}(k) - (f^{\text{out}})^{-1} y_{[n]}(k) \right)^2, \tag{6}$$

where $\lambda < 1$ is the *forgetting factor* and $y_{[n]}(k)$ is the model output that would be obtained at time $k$ when a network with the current output weights $\mathbf{w}_{\text{squares}}^{\text{out}}(n)$ would be employed at all times $k = 1, \ldots, n$.

There are many variants of RLS algorithms minimizing (6), differing in their trade-offs between computational cost, simplicity, and numerical stability. I use a "vanilla" version, which is detailed out in Table 12.1 in [2] and in the web tutorial package accompanying this paper.

Two parameters characterise the tracking performance of an RLS algorithm: the misadjustment $\mathcal{M}$ and the convergence time constant $\tau$. The misadjustment gives the ratio between the excess MSE (or excess NMSE) incurred by the fluctuations of the adaptation process, and the optimal steady-state MSE that would be obtained in the limit of offline-training on infinite stationary training data. For instance, a misadjustment of $\mathcal{M} = 0.3$ means that the tracking error of the adaptive algorithm in a steady-state situation exceeds the theoretically achievable optimum (with same tap weight vector length) by 30 %. The time constant $\tau$ associated with an RLS algorithm determines the exponent of the MSE convergence, $e^{-n/\tau}$. For example, $\tau = 200$ would imply an excess MSE reduction by $1/e$ every 200 steps. Misadjustment and convergence exponent are related to the forgetting factor and the tap-vector length through

$$\mathcal{M} = N \frac{1 - \lambda}{1 + \lambda} \quad \text{and} \quad \tau \approx \frac{1}{1 - \lambda}. \tag{7}$$

## 4.2 Case study: RLS-ESN for our 10th-order system

Eqns. (7) can be used to predict/design the tracking characteristics of a RLS-powered ESN. I will demonstrate this with the 10th-order system (5). I re-use the same augmented 100-unit ESN, but now determine its $2N + 2$ output weight vector online with RLS. Setting $\lambda = 0.995$, and considering $N = 202$, Eqns. (7) yield a misadjustment of $\mathcal{M} = 0.5$ and a time constant $\tau \approx 200$. Since the asymptotically optimal NMSE is approximately the NMSE of the offline-trained network, namely, NMSE $\approx 0.032$, the misadjustment $\mathcal{M} = 0.5$ lets us expect a NMSE of $0.032 \times 150\% \approx 0.048$ for the online adaptation after convergence. The time constant $\tau \approx 200$ makes us expect NMSE convergence to the expected asymptotic NMSE by a factor of $1/e$ every 200 steps.

*Training data.* Experiments with the system (5) revealed that the system sometimes explodes when driven with i.i.d. input from $[0, 0.5]$. To bound outputs, I wrapped the r.h.s. of (5) with a tanh. Furthermore, I replaced the original constants $0.3, 0.05, 1.5, 0.1$ by free parameters $\alpha, \beta, \gamma, \delta$, to obtain

$$d(n+1) = \tanh\left(\alpha\, d(n) + \beta\, d(n) \left[\sum_{i=0}^{9} d(n-i)\right] + \gamma\, u(n-9)\, u(n) + \delta\right). \quad (8)$$

This system was run for 10000 steps with an i.i.d. teacher input from $[0, 0.5]$. Every 2000 steps, $\alpha, \beta, \gamma, \delta$ were assigned new random values taken from a $\pm 50\,\%$ interval around the respective original constants. Fig. 2A shows the resulting teacher output sequence, which clearly shows transitions between different "episodes" every 2000 steps.

*Running the RLS-ENS algorithm.* The ENS was started from zero state and with a zero augmented output weight vector. It was driven by the teacher input, and a noise of size 0.0001 was inserted into the state update, as in the offline training. The RLS algorithm (with forgetting factor 0.995) was initialized according to the prescriptions given in [2] and then run together with the network updates, to compute from the augmented input+network states $\mathbf{x}(n) = (u(n), x_1(n), \ldots, x_N(n), u^2(n), x_1^2(n), \ldots, x_N^2(n))$ a sequence of augmented output weight vectors $\mathbf{w}_{\text{squares}}^{\text{out}}(n)$. These output weight vectors were used to calculate a network output $y(n) = \tanh(\mathbf{w}_{\text{squares}}^{\text{out}}(n), \mathbf{x}(n))$.

*Results.* From the resulting length-10000 sequences of desired outputs $d(n)$ and network productions $y(n)$, NMSE's were numerically estimated from averaging within subsequent length-100 blocks. Fig. 2B gives a logarithmic plot.

In the last three episodes, the exponential NMSE convergence after each episode onset disruption is clearly recognizable. Also the convergence speed matches the predicted time constant, as revealed by the $\tau = 200$ slope line inserted in Fig. 2B.

The dotted horizontal line in Fig. 2B marks the NMSE of the offline-trained ESN described in the previous section. Surprisingly, after convergence, the online-NMSE is lower than the offline NMSE. This can be explained through the IIR (autoregressive) nature of the system (5) resp. (8), which incurs long-term correlations in the signal $d(n)$, or in other words, a nonstationarity of the signal in the timescale of the correlation lengthes, even with fixed parameters $\alpha, \beta, \gamma, \delta$. This medium-term nonstationarity compromises the performance of the offline algorithm, but the online adaptation can to a certain degree follow this nonstationarity.

Fig. 2C is a logarithmic plot of the development of the mean absolute output weight size. It is apparent that after starting from zero, there is an initial exponential growth of absolute values of the output weights, until a stabilization at a size of about 1000, whereafter the NMSE develops a regular pattern (Fig. 2B).

Finally, Fig. 2D shows an overlay of $d(n)$ (solid) with $y(n)$ (dotted) of the last 100 steps in the experiment, visually demonstrating the precision after convergence.

*A note on noise and stability.* Standard offline training of ESNs yields output weights whose absolute size depends on the noise inserted into the network during training: the larger the noise, the smaller the mean output weights (extensive discussion in [5]). In online training, a similar inverse correlation between output weight size (after settling on plateau) and noise size can be observed. When the online learning experiment was done otherwise identically but without noise insertion, weights grew so large that the RLS algorithm entered a region of numerical

instability. Thus, the noise term is crucial here for numerical stability, a condition familiar from EKF-based RNN training schemes [3], which are computationally closely related to RLS.

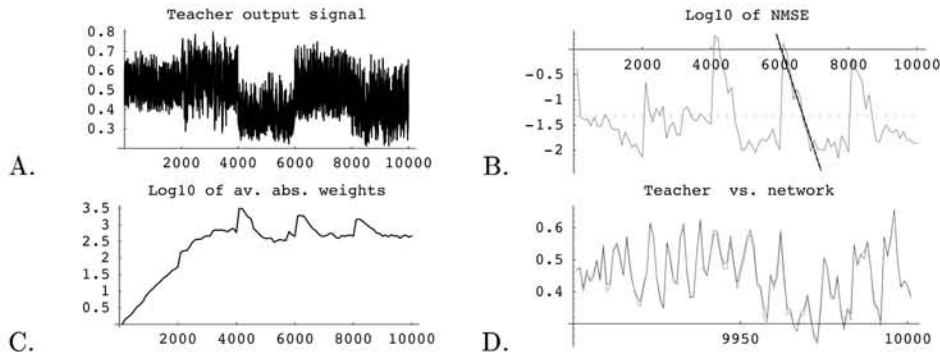

Figure 2: A. Teacher output. B. NMSE with predicted baseline and slopeline. C. Development of weights. D. Last 100 steps: desired (solid) and network-predicted (dashed) signal. For details see text.

## 5    Discussion

Several of the well-known error-gradient-based RNN training algorithms can be used for online weight adaptation. The update costs per time step in the most efficient of those algorithms (overview in [1]) are $O(N^2)$, where $N$ is network size. Typically, standard approaches train small networks (order of $N = 20$), whereas ESN typically relies on large networks for precision (order of $N = 100$). Thus, the RLS-based ESN online learning algorithm is typically more expensive than standard techniques. However, this drawback might be compensated by the following properties of RLS-ESN:

- Simplicity of design and implementation; robust behavior with little need for learning parameter hand-tuning.
- Custom-design of RLS-ESNs with prescribed tracking parameters, transferring well-understood linear systems methods to nonlinear systems.
- Systems with long-lasting short-term memory can be learnt. Exploitable ESN memory spans grow with network size (analysis in [6]). Consider the 30th order system $d(n+1) = \tanh(0.2\,d(n) + 0.04\,d(n)\left[\sum_{i=0}^{2} 9d(n-i)\right] + 1.5\,u(n-29)\,u(n) + 0.001)$. It was learnt by a 400-unit augmented adaptive ESN with a test NMSE of 0.0081. The 51-th (!) order system $y(n+1) = u(n-10)\,u(n-50)$ was learnt offline by a 400-unit augmented ESN with a NMSE of 0.21[3].

All in all, on the kind of tasks considered in above, adaptive (augmented) ESNs reach a similar level of precision as today's most refined gradient-based techniques. A given level of precision is attained in ESN vs. gradient-based techniques with a similar number of trainable weights (D. Prokhorov, private communication). Because gradient-based techniques train every connection weight in the RNN, whereas

ESNs train only the output weights, the numbers of units of similarly performing standard RNNs vs. ESNs relate as $N$ to $N^2$. Thus, RNNs are more compact than equivalent ESNs. However, when working with ESNs, for each new trained output signal one can re-use the same "reservoir", adding only $N$ new connections and weights. This has for instance been exploited for robots in the AIS institute by simultaneously training multiple feature detectors from a single "reservoir" [4]. In this circumstance, with a growing number of simultaneously required outputs, the requisite net model sizes for ESNs vs. traditional RNNs become asymptotically equal. The size disadvantage of ESNs is further balanced by much faster offline training, greater simplicity, and the general possibility to exploit linear-systems expertise for nonlinear adaptive modeling.

**Acknowledgments** The results described in this paper were obtained while I worked at the Fraunhofer AIS Institute. I am greatly indebted to Thomas Christaller for unfaltering support. Wolfgang Maass and Danil Prokhorov contributed motivating discussions and valuable references. An international patent application for the ESN technique was filed on October 13, 2000 (PCT/EP01/11490).

## Footnotes

[1]All algorithms and calculations described in this article are con-

[2]The authors miscalculated their NMSE because they used a formula for zero-mean signals. I re-calculated the value $\text{NMSE}_{\text{train}} \approx 0.241$ from their reported best (miscalculated) NMSE of 0.015. The larger value agrees with the plots supplied in that paper.

[3]See Mathematica notebook for details.

# References

[1] A.F. Atiya and A.G. Parlos. New results on recurrent network training: Unifying the algorithms and accelerating convergence. *IEEE Trans. Neural Networks*, 11(3):697–709, 2000.

[2] B. Farhang-Boroujeny. *Adaptive Filters: Theory and Applications*. Wiley, 1998.

[3] L.A. Feldkamp, D.V. Prokhorov, C.F. Eagen, and F. Yuan. Enhanced multistream Kalman filter training for recurrent neural networks. In J.A.K. Suykens and J. Vandewalle, editors, *Nonlinear Modeling: Advanced Black-Box Techniques*, pages 29–54. Kluwer, 1998.

[4] J. Hertzberg, H. Jaeger, and F. Schönherr. Learning to ground fact symbols in behavior-based robots. In F. van Harmelen, editor, *Proc. 15th Europ. Conf. on Art. Int. (ECAI 02)*, pages 708–712. IOS Press, Amsterdam, 2002.

[5] H. Jaeger. The "echo state" approach to analysing and training recurrent neural networks. GMD Report 148, GMD - German National Research Institute for Computer Science, 2001. http://www.gmd.de/People/Herbert.Jaeger/Publications.html.

[6] H. Jaeger. Short term memory in echo state networks. GMD-Report 152, GMD - German National Research Institute for Computer Science, 2002. http://www.gmd.de/People/Herbert.Jaeger/Publications.html.

[7] H. Jaeger. Tutorial on training recurrent neural networks, covering BPPT, RTRL, EKF and the echo state network approach. GMD Report 159, Fraunhofer Institute AIS, 2002.

[8] W. Maass, T. Natschlaeger, and H. Markram. Real-time computing without stable states: A new framework for neural computation based on perturbations. http://www.cis.tugraz.at/igi/maass/psfiles/LSM-v106.pdf, 2002.

[9] W. Maass, Th. Natschläger, and H. Markram. A model for real-time computation in generic neural microcircuits. In S. Becker, S. Thrun, and K. Obermayer, editors, *Advances in Neural Information Processing System 15 (Proc. NIPS 2002)*. MIT Press, 2002.
